# Optimal kernel choice for large-scale two-sample tests

**Arthur Gretton,**[1,3] **Bharath Sriperumbudur,**[1] **Dino Sejdinovic,**[1] **Heiko Strathmann**[2]
[1]Gatsby Unit and [2]CSD, CSML, UCL, UK; [3]MPI for Intelligent Systems, Germany
{arthur.gretton,bharat.sv,dino.sejdinovic,heiko.strathmann}@gmail

| **Sivaraman Balakrishnan** | **Massimiliano Pontil** | **Kenji Fukumizu** |
|:---:|:---:|:---:|
| LTI, CMU, USA | CSD, CSML, UCL, UK | ISM, Japan |
| sbalakri@cs.cmu.edu | m.pontil@cs.ucl.ac.uk | fukumizu@ism.ac.jp |

## Abstract

Given samples from distributions $p$ and $q$, a two-sample test determines whether to reject the null hypothesis that $p = q$, based on the value of a test statistic measuring the distance between the samples. One choice of test statistic is the maximum mean discrepancy (MMD), which is a distance between embeddings of the probability distributions in a reproducing kernel Hilbert space. The kernel used in obtaining these embeddings is critical in ensuring the test has high power, and correctly distinguishes unlike distributions with high probability. A means of parameter selection for the two-sample test based on the MMD is proposed. For a given test level (an upper bound on the probability of making a Type I error), the kernel is chosen so as to maximize the test power, and minimize the probability of making a Type II error. The test statistic, test threshold, and optimization over the kernel parameters are obtained with cost linear in the sample size. These properties make the kernel selection and test procedures suited to data streams, where the observations cannot all be stored in memory. In experiments, the new kernel selection approach yields a more powerful test than earlier kernel selection heuristics.

## 1 Introduction

The two sample problem addresses the question of whether two independent samples are drawn from the same distribution. In the setting of statistical hypothesis testing, this corresponds to choosing whether to reject the null hypothesis $\mathcal{H}_0$ that the generating distributions $p$ and $q$ are the same, vs. the alternative hypothesis $\mathcal{H}_A$ that distributions $p$ and $q$ are different, given a set of independent observations drawn from each.

A number of recent approaches to two-sample testing have made use of mappings of the distributions to a reproducing kernel Hilbert space (RKHS); or have sought out RKHS functions with large amplitude where the probability mass of $p$ and $q$ differs most [8, 10, 15, 17, 7]. The most straightforward test statistic is the norm of the difference between distribution embeddings, and is called the maximum mean discrepancy (MMD). One difficulty in using this statistic in a hypothesis test, however, is that the MMD depends on the choice of the kernel. If we are given a family of kernels, we obtain a different value of the MMD for each member of the family, and indeed for any positive definite linear combination of the kernels. When a radial basis function kernel (such as the Gaussian kernel) is used, one simple choice is to set the kernel width to the median distance between points in the aggregate sample [8, 7]. While this is certainly straightforward, it has no guarantees of optimality. An alternative heuristic is to choose the kernel that maximizes the test statistic [15]: in experiments, this was found to reliably outperform the median approach. Since the MMD returns a smooth RKHS function that minimizes classification error under linear loss, then maximizing the

MMD corresponds to minimizing this classification error under a smoothness constraint. If the statistic is to be applied in hypothesis testing, however, then this choice of kernel does not explicitly address the question of test performance.

We propose a new approach to kernel choice for hypothesis testing, which explicitly optimizes the performance of the hypothesis test. Our kernel choice minimizes Type II error (the probability of wrongly accepting $\mathcal{H}_0$ when $p \neq q$), given an upper bound on Type I error (the probability of wrongly rejecting $\mathcal{H}_0$ when $p = q$). This corresponds to optimizing the asymptotic relative efficiency in the sense of Hodges and Lehmann [13, Ch. 10]. We address the case of the linear time statistic in [7, Section 6], for which both the test statistic and the parameters of the null distribution can be computed in $O(n)$, for sample size $n$. This has a higher variance at a given $n$ than the U-statistic estimate costing $O(n^2)$ used in [8, 7], since the latter is the minimum variance unbiased estimator. Thus, we would use the quadratic time statistic in the "limited data, unlimited time" scenario, as it extracts the most possible information from the data available. The linear time statistic is used in the "unlimited data, limited time" scenario, since it is the cheapest statistic that still incorporates each datapoint: it does not require the data to be stored, and is thus appropriate for analyzing data streams. As a further consequence of the streaming data setting, we learn the kernel parameter on a separate sample to the sample used in testing; i.e., unlike the classical testing scenario, we use a training set to learn the kernel parameters. An advantage of this setting is that our null distribution remains straightforward, and the test threshold can be computed without a costly bootstrap procedure.

We begin our presentation in Section 2 with a review of the maximum mean discrepancy, its linear time estimate, and the associated asymptotic distribution and test. In Section 3 we describe a criterion for kernel choice to maximize the Hodges and Lehmann asymptotic relative efficiency. We demonstrate the convergence of the empirical estimate of this criterion when the family of kernels is a linear combination of base kernels (with non-negative coefficients), and of the kernel coefficients themselves. In Section 4, we provide an optimization procedure to learn the kernel weights. Finally, in Section 5, we present experiments, in which we compare our kernel selection strategy with the approach of simply maximizing the test statistic subject to various constraints on the coefficients of the linear combination; and with a cross-validation approach, which follows from the interpretation of the MMD as a classifier. We observe that a principled kernel choice for testing outperforms competing heuristics, including the previous best-performing heuristic in [15]. A Matlab implementation is available at: `www.gatsby.ucl.ac.uk/` $\sim$ `gretton/adaptMMD/adaptMMD.htm`

## 2   Maximum mean discrepancy, and a linear time estimate

We begin with a brief review of kernel methods, and of the maximum mean discrepancy [8, 7, 14]. We then describe the family of kernels over which we optimize, and the linear time estimate of the MMD.

### 2.1   MMD for a family of kernels

Let $\mathcal{F}_k$ be a reproducing kernel Hilbert space (RKHS) defined on a topological space $\mathcal{X}$ with reproducing kernel $k$, and $p$ a Borel probability measure on $\mathcal{X}$. The *mean embedding* of $p$ in $\mathcal{F}_k$ is a unique element $\mu_k(p) \in \mathcal{F}_k$ such that $\mathbf{E}_{x \sim p} f(x) = \langle f, \mu_k(p) \rangle_{\mathcal{F}_k}$ for all $f \in \mathcal{F}_k$ [4]. By the Riesz representation theorem, a sufficient condition for the existence of $\mu_k(p)$ is that $k$ be Borel-measurable and $\mathbf{E}_{x \sim p} k^{1/2}(x, x) < \infty$. We assume $k$ is a bounded continuous function, hence this condition holds for all Borel probability measures. The maximum mean discrepancy (MMD) between Borel probability measures $p$ and $q$ is defined as the RKHS-distance between the mean embeddings of $p$ and $q$. An expression for the squared MMD is thus

$$\eta_k(p, q) = \|\mu_k(p) - \mu_k(q)\|^2_{\mathcal{F}_k} = \mathbf{E}_{xx'} k(x, x') + \mathbf{E}_{yy'} k(y, y') - 2\mathbf{E}_{xy} k(x, y), \qquad (1)$$

where $x, x' \overset{i.i.d.}{\sim} p$ and $y, y' \overset{i.i.d.}{\sim} q$. By introducing

$$h_k(x, x', y, y') = k(x, x') + k(y, y') - k(x, y') - k(x', y),$$

we can write

$$\eta_k(p, q) = \mathbf{E}_{xx'yy'} h_k(x, x', y, y') =: \mathbf{E}_v h_k(v), \qquad (2)$$

where we have defined the random vector $v := [x, x', y, y']$. If $\mu_k$ is an injective map, then $k$ is said to be a characteristic kernel, and the MMD is a metric on the space of Borel probability measures, i.e., $\eta_k(p, q) = 0$ iff $p = q$ [16]. The Gaussian kernels used in the present work are characteristic.

Our goal is to select a kernel for hypothesis testing from a particular family $\mathcal{K}$ of kernels, which we now define. Let $\{k_u\}_{u=1}^{d}$ be a set of positive definite functions $k_u : \mathcal{X} \times \mathcal{X} \to \mathbb{R}$. Let

$$\mathcal{K} := \left\{ k : k = \sum_{u=1}^{d} \beta_u k_u, \sum_{u=1}^{d} \beta_u = D, \beta_u \geq 0, \forall u \in \{1, \dots, d\} \right\} \qquad (3)$$

for some $D > 0$, where the constraint on the sum of coefficients is needed for the consistency proof (see Section 3). Each $k \in \mathcal{K}$ is associated uniquely with an RKHS $\mathcal{F}_k$, and we assume the kernels are bounded, $|k_u| \leq K, \forall u \in \{1, \dots, d\}$. The squared MMD becomes

$$\eta_k(p, q) = \|\mu_k(p) - \mu_k(q)\|_{\mathcal{F}_k}^2 = \sum_{u=1}^{d} \beta_u \eta_u(p, q),$$

where $\eta_u(p, q) := \mathbf{E}_v h_u(v)$. It is clear that if every kernel $k_u, u \in \{1, \dots, d\}$, is characteristic and at least one $\beta_u > 0$, then $k$ is characteristic. Where there is no ambiguity, we will write $\eta_u := \eta_u(p, q)$ and $\mathbf{E}h_u := \mathbf{E}_v h_u(v)$. We denote $h = (h_1, h_2, \dots, h_d)^{\top} \in \mathbb{R}^{d \times 1}$, $\beta = (\beta_1, \beta_2, \dots, \beta_d)^{\top} \in \mathbb{R}^{d \times 1}$, and $\eta = (\eta_1, \eta_2, \dots, \eta_d)^{\top} \in \mathbb{R}^{d \times 1}$. With this notation, we may write

$$\eta_k(p, q) = \mathbf{E}(\beta^{\top} h) = \beta^{\top} \eta.$$

## 2.2 Empirical estimate of the MMD, asymptotic distribution, and test

We now describe an empirical estimate of the maximum mean discrepancy, given i.i.d. samples $X := \{x_1, \dots, x_n\}$ and $Y := \{y_1, \dots, y_n\}$ from $p$ and $q$, respectively. We use the linear time estimate of [7, Section 6], for which both the test statistic and the parameters of the null distribution can be computed in time $O(n)$. This has a higher variance at a given $n$ than a U-statistic estimate costing $O(n^2)$, since the latter is the minimum variance unbiased estimator [13, Ch. 5]. That said, it was observed experimentally in [7, Section 8.3] that the linear time statistic yields better performance at a given computational cost than the quadratic time statistic, when sufficient data are available (bearing in mind that consistent estimates of the null distribution in the latter case are computationally demanding [9]). Moreover, the linear time statistic does not require the sample to be stored in memory, and is thus suited to data streaming contexts, where a large number of observations arrive in sequence.

The linear time estimate of $\eta_k(p, q)$ is defined in [7, Lemma 14]: assuming for ease of notation that $n$ is even,

$$\check{\eta}_k = \frac{2}{n} \sum_{i=1}^{n/2} h_k(v_i), \qquad (4)$$

where $v_i := [x_{2i-1}, x_{2i}, y_{2i-1}, y_{2i}]$ and $h_k(v_i) := k(x_{2i-1}, x_{2i}) + k(y_{2i-1}, y_{2i}) - k(x_{2i-1}, y_{2i}) - k(x_{2i}, y_{2i-1})$; this arrangement of the samples ensures we get an expectation over independent variables as in (2) with cost $O(n)$. We use $\check{\eta}_k$ to denote the empirical statistic computed over the samples being tested, to distinguish it from the training sample estimate $\hat{\eta}_k$ used in selecting the kernel. Given the family of kernels $\mathcal{K}$ in (3), this can be written $\check{\eta}_k = \beta^{\top} \check{\eta}$, where we again use the convention $\check{\eta} = (\check{\eta}_1, \check{\eta}_2, \dots, \check{\eta}_d)^{\top} \in \mathbb{R}^{d \times 1}$. The statistic $\check{\eta}_k$ has expectation zero under the null hypothesis $\mathcal{H}_0$ that $p = q$, and has positive expectation under the alternative hypothesis $\mathcal{H}_A$ that $p \neq q$.

Since $\check{\eta}_k$ is a straightforward average of independent random variables, its asymptotic distribution is given by the central limit theorem (e.g. [13, Section 1.9]). From [7, corollary 16], under the assumption $0 < \mathbf{E}(h_k^2) < \infty$ (which is true for bounded continuous $k$),

$$n^{1/2} (\check{\eta}_k - \eta_k(p, q)) \xrightarrow{D} \mathcal{N}(0, 2\sigma_k^2), \qquad (5)$$

where the factor of two arises since the average is over $n/2$ terms, and

$$\sigma_k^2 = \mathbf{E}_v h_k^2(v) - [\mathbf{E}_v (h_k(v))]^2. \qquad (6)$$

Unlike the case of a quadratic time statistic, the null and alternative distributions differ only in mean; by contrast, the quadratic time statistic has as its null distribution an infinite weighted sum of $\chi^2$ variables [7, Section 5], and a Gaussian alternative distribution.

To obtain an estimate of the variance based on the samples $X, Y$, we will use an expression derived from the U-statistic of [13, p. 173] (although as earlier, we will express this as a simple average so as to compute it in linear time). The population variance can be written

$$\sigma_k^2 = \mathbf{E}_v h_k^2(v) - \mathbf{E}_{v,v'}(h_k(v)h_k(v')) = \frac{1}{2}\mathbf{E}_{v,v'}(h_k(v) - h_k(v'))^2.$$

Expanding in terms of the kernel coefficients $\beta$, we get

$$\sigma_k^2 := \beta^\top Q_k \beta,$$

where $Q_k := \text{cov}(h)$ is the covariance matrix of $h$. A linear time estimate for the variance is

$$\check{\sigma}_k^2 = \beta^\top \check{Q}_k \beta, \quad \text{where} \quad \left(\check{Q}_k\right)_{uu'} = \frac{4}{n}\sum_{i=1}^{n/4} h_{\Delta,u}(w_i)h_{\Delta,u'}(w_i), \tag{7}$$

and $w_i := [v_{2i-1},\ v_{2i}],^1$ $h_{\Delta,k}(w_i) := h_k(v_{2i-1}) - h_k(v_{2i})$.

We now address the construction of a hypothesis test. We denote by $\Phi$ the CDF of a standard Normal random variable $\mathcal{N}(0, 1)$, and by $\Phi^{-1}$ the inverse CDF. From (5), a test of asymptotic level $\alpha$ using the statistic $\check{\eta}_k$ will have the threshold

$$t_{k,\alpha} = n^{-1/2}\sigma_k\sqrt{2}\Phi^{-1}(1-\alpha), \tag{8}$$

bearing in mind the asymptotic distribution of the test statistic, and that $\eta_k(p,p) = 0$. This threshold is computed empirically by replacing $\sigma_k$ with its estimate $\check{\sigma}_k$ (computed using the data being tested), which yields a test of the desired asymptotic level.

The asymptotic distribution (5) holds only when the kernel is fixed, and does not depend on the sample $X, Y$. If the kernel were a function of the data, then a test would require large deviation probabilities over the supremum of the Gaussian process indexed by the kernel parameters (e.g. [1]). In practice, the threshold would be computed via a bootstrap procedure, which has a high computational cost. Instead, we set aside a portion of the data to learn the kernel (the "training data"), and use the remainder to construct a test using the learned kernel parameters.

## 3 Choice of kernel

The choice of kernel will affect both the test statistic itself, (4), and its asymptotic variance, (6). Thus, we need to consider how these statistics determine the power of a test with a given level $\alpha$ (the upper bound on the Type I error). We consider the case where $p \neq q$. A Type II error occurs when the random variable $\check{\eta}_k$ falls below the threshold $t_{k,\alpha}$ defined in (8). The asymptotic probability of a Type II error is therefore

$$P(\check{\eta}_k < t_{k,\alpha}) = \Phi\left(\Phi^{-1}(1-\alpha) - \frac{\eta_k(p,q)\sqrt{n}}{\sigma_k\sqrt{2}}\right).$$

As $\Phi$ is monotonic, the Type II error probability will decrease as the ratio $\eta_k(p,q)\sigma_k^{-1}$ increases. Therefore, the kernel minimizing this error probability is

$$k_* = \arg\sup_{k \in \mathcal{K}} \eta_k(p,q)\sigma_k^{-1}, \tag{9}$$

with the associated test threshold $t_{k_*,\alpha}$. In practice, we do not have access to the population estimates $\eta_k(p,q)$ and $\sigma_k$, but only their empirical estimates $\hat{\eta}_k, \hat{\sigma}_k$ from $m$ pairs of training points $(x_i, y_i)$ (this training sample must be independent of the sample used to compute the test parameters $\check{\eta}_k, \check{\sigma}_k$). We therefore estimate $t_{k_*,\alpha}$ by a regularized empirical estimate $t_{\hat{k}_*,\alpha}$, where

$$\hat{k}_* = \arg\sup_{k \in \mathcal{K}} \hat{\eta}_k (\hat{\sigma}_{k,\lambda})^{-1},$$

and we define the regularized standard deviation $\hat{\sigma}_{k,\lambda} = \sqrt{\beta^\top \left(\hat{Q} + \lambda_m I\right)\beta} = \sqrt{\hat{\sigma}_k^2 + \lambda_m \|\beta\|_2^2}$.

The next theorem shows the convergence of $\sup_{k \in \mathcal{K}} \hat{\eta}_k \left(\hat{\sigma}_{k,\lambda}\right)^{-1}$ to $\sup_{k \in \mathcal{K}} \eta_k(p,q)\sigma_k^{-1}$, and of $\hat{k}_*$ to $k_*$, for an appropriate schedule of decrease for $\lambda_m$ with increasing $m$.

**Theorem 1.** *Let $\mathcal{K}$ be defined as in* (3). *Assume $\sup_{k \in \mathcal{K}, x, y \in \mathcal{X}} |k(x,y)| < K$ and $\sigma_k$ is bounded away from zero. Then if $\lambda_m = \Theta\left(m^{-1/3}\right)$,*

$$\left| \sup_{k \in \mathcal{K}} \hat{\eta}_k \hat{\sigma}_{k,\lambda}^{-1} - \sup_{k \in \mathcal{K}} \eta_k \sigma_k^{-1} \right| = O_P\left(m^{-1/3}\right) \qquad \text{and} \qquad \hat{k}_* \xrightarrow{P} k_*.$$

*Proof.* Recall from the definition of $\mathcal{K}$ that $\|\beta\|_1 = D$, and that $\|\beta\|_2 \le \|\beta\|_1$ and $\|\beta\|_1 \le \sqrt{d}\,\|\beta\|_2$ [11, Problem 3 p. 278], hence $\|\beta\|_2 \ge Dd^{-1/2}$. We begin with the bound

$$\left| \sup_{k \in \mathcal{K}} \hat{\eta}_k \hat{\sigma}_{k,\lambda}^{-1} - \sup_{k \in \mathcal{K}} \eta_k \sigma_k^{-1} \right| \le \sup_{k \in \mathcal{K}} \left| \hat{\eta}_k \hat{\sigma}_{k,\lambda}^{-1} - \eta_k \sigma_k^{-1} \right|$$

$$\le \sup_{k \in \mathcal{K}} \left| \hat{\eta}_k \hat{\sigma}_{k,\lambda}^{-1} - \eta_k \sigma_{k,\lambda}^{-1} \right| + \sup_{k \in \mathcal{K}} \left| \eta_k \sigma_{k,\lambda}^{-1} - \eta_k \sigma_k^{-1} \right|$$

$$\le \sup_{k \in \mathcal{K}} \left(\hat{\sigma}_k^2 + \|\beta\|_2^2 \lambda_m\right)^{-1/2} |\hat{\eta}_k - \eta_k| + \sup_{k \in \mathcal{K}} \eta_k \left| \frac{\hat{\sigma}_{k,\lambda} - \sigma_{k,\lambda}}{\hat{\sigma}_{k,\lambda} \sigma_{k,\lambda}} \right| + \sup_{k \in \mathcal{K}} \frac{\eta_k}{\sigma_k} \left| \frac{\sigma_{k,\lambda}^2 - \sigma_k^2}{\sigma_{k,\lambda}\left(\sigma_{k,\lambda} + \sigma_k\right)} \right|$$

$$\le \frac{C_1 \sqrt{d}}{D \sqrt{\lambda_m}} \sup_{k \in \mathcal{K}} |\hat{\eta}_k - \eta_k| + \sup_{k \in \mathcal{K}} \eta_k \left| \frac{\hat{\sigma}_{k,\lambda} - \sigma_{k,\lambda}}{\left(\sigma_k^2 \hat{\sigma}_k^2 + \|\beta\|_2^2 \lambda_m \left(\sigma_k^2 + \hat{\sigma}_k^2\right) + \|\beta\|_2^2 \lambda_m^2\right)^{1/2}} \right|$$

$$\qquad + \sup_{k \in \mathcal{K}} \frac{\eta_k}{\sigma_k} \left( \frac{\|\beta\|_2^2 \lambda_m}{\|\beta\|_2^2 \lambda_m + \sigma_k^2} \right)$$

$$\le \frac{\sqrt{d}}{D \sqrt{\lambda_m}} \left( C_1 \sup_{k \in \mathcal{K}} |\hat{\eta}_k - \eta_k| + C_2 \sup_{k \in \mathcal{K}} |\hat{\sigma}_{k,\lambda} - \sigma_{k,\lambda}| \right) + C_3 D^2 \lambda_m,$$

where constants $C_1$, $C_2$, and $C_3$ follow from the boundedness of $\sigma_k$ and $\eta_k$. The the first result in the theorem follows from $\sup_{k \in \mathcal{K}} |\hat{\eta}_k - \eta_k| = O_P(m^{-1/2})$ and $\sup_{k \in \mathcal{K}} |\hat{\sigma}_{k,\lambda} - \sigma_{k,\lambda}| = O_P(m^{-1/2})$, which are proved using McDiarmid's Theorem [12] and results from [3]: see Appendix A of the supplementary material.

**Convergence of $\hat{k}_*$ to $k_*$:** For $k \in \mathcal{K}$ defined in (3), we show in Section 4 that $\hat{k}_*$ and $k_*$ are unique optimizers of $\hat{\eta}_k \hat{\sigma}_{k,\lambda}^{-1}$ and $\eta_k \sigma_k^{-1}$, respectively. Since $\sup_{k \in \mathcal{K}} \frac{\hat{\eta}_k}{\hat{\sigma}_{k,\lambda}} \xrightarrow{P} \sup_{k \in \mathcal{K}} \frac{\eta_k}{\sigma_k}$, the result follows from [18, Corollary 3.2.3(i)]. $\qquad\square$

We remark that other families of kernels may be worth considering, besides $\mathcal{K}$. For instance, we could use a family of RBF kernels with continuous bandwidth parameter $\theta \ge 0$. We return to this point in the conclusions (Section 6).

## 4  Optimization procedure

We wish to select kernel $k = \sum_{u=1}^d \hat{\beta}_u^* k_u \in \mathcal{K}$ that maximizes the ratio $\hat{\eta}_k / \hat{\sigma}_{k,\lambda}$. We perform this optimization over training data, then use the resulting parameters $\hat{\beta}^*$ to construct a hypothesis test on the data to be tested (which must be independent of the training data, and drawn from the same $p, q$). As discussed in Section 2.2, this gives us the test threshold without requiring a bootstrap procedure. Recall from Sections 2.2 and 3 that $\hat{\eta}_k = \beta^\top \hat{\eta}$, and $\hat{\sigma}_{k,\lambda} = \sqrt{\beta^\top \left(\hat{Q} + \lambda_m I\right)\beta}$, where $\hat{Q}$ is a linear-time empirical estimate of the covariance matrix $\mathrm{cov}(h)$. Since the objective $\alpha(\beta; \hat{\eta}, \hat{Q}) := \left(\beta^\top \hat{\eta}\right)\left(\beta^\top \left(\hat{Q} + \lambda_m I\right)\beta\right)^{-1/2}$ is a homogenous function of order zero in $\beta$, we can omit the constraint $\|\beta\|_1 = D$, and set

$$\hat{\beta}^* = \arg\max_{\beta \succeq 0} \alpha(\beta; \hat{\eta}, \hat{Q}). \tag{10}$$

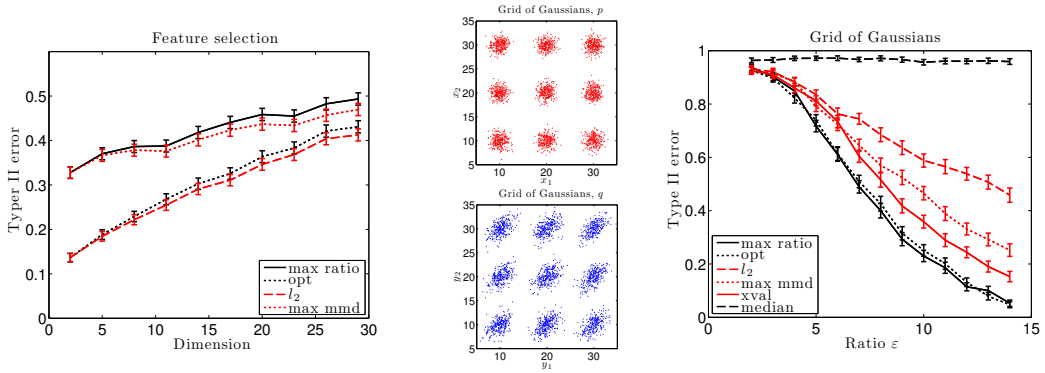

Figure 1: **Left:** Feature selection results, Type II error vs number of dimensions, average over 5000 trials, $m = n = 10^4$. **Centre:** $3 \times 3$ Gaussian grid, samples from $p$ and $q$. **Right:** Gaussian grid results, Type II error vs $\varepsilon$, the eigenvalue ratio for the covariance of the Gaussians in $q$; average over 1500 trials, $m = n = 10^4$. The asymptotic test level was $\alpha = 0.05$ in both experiments. Error bars give the 95% Wald confidence interval.

If $\hat{\eta}$ has at least one positive entry, there exists $\beta \succeq 0$ such that $\alpha(\beta; \hat{\eta}, \hat{Q}) > 0$. Then clearly, $\alpha(\hat{\beta}^*; \hat{\eta}, \hat{Q}) > 0$, so we can write $\hat{\beta}^* = \arg\max_{\beta \succeq 0} \alpha^2(\beta; \hat{\eta}, \hat{Q})$. In this case, the problem (10) becomes equivalent to a (convex) quadratic program with a unique solution, given by

$$\min\{\beta^\top \left(\hat{Q} + \lambda_m I\right) \beta : \beta^\top \hat{\eta} = 1, \beta \succeq 0\}. \tag{11}$$

Under the alternative hypothesis, we have $\eta_u > 0$, $\forall u \in \{1, \ldots, d\}$, so the same reasoning can be applied to the population version of the optimization problem, i.e., to $\beta^* = \arg\max_{\beta \succeq 0} \alpha(\beta; \eta, \text{cov}(h))$, which implies the optimizer $\beta^*$ is unique. In the case where no entries in $\hat{\eta}$ are positive, we obtain maximization of a quadratic form subject to a linear constraint,

$$\max\{\beta^\top \left(\hat{Q} + \lambda_m I\right) \beta : \beta^\top \hat{\eta} = -1, \beta \succeq 0\}.$$

While this problem is somewhat more difficult to solve, in practice its exact solution is irrelevant to the Type II error performance of the proposed two-sample test. Indeed, since all of the squared MMD estimates calculated on the training data using each of the base kernels are negative, it is unlikely the statistic computed on the data used for the test will exceed the (always positive) threshold. Therefore, when no entries in $\hat{\eta}$ are positive, we (arbitrarily) select a single base kernel $k_u$ with largest $\hat{\eta}_u / \hat{\sigma}_{u,\lambda}$.

The key component of the optimization procedure is the quadratic program in (11). This problem can be solved by interior point methods, or, if the number of kernels $d$ is large, we could use proximal-gradient methods. In this case, an $\epsilon$-minimizer can be found in $O(d^2/\sqrt{\epsilon})$ time. Therefore, the overall computational cost of the proposed test is linear in the number of samples, and quadratic in the number of kernels.

## 5 Experiments

We compared our kernel selection strategy to alternative approaches, with a focus on challenging problems that benefit from careful kernel choice. In our first experiment, we investigated a synthetic data set for which the best kernel in the family $\mathcal{K}$ of linear combinations in (3) outperforms the best individual kernel from the set $\{k_u\}_{u=1}^d$ . Here $p$ was a zero mean Gaussian with unit covariance, and $q$ was a mixture of two Gaussians with equal weight, one with mean 0.5 in the first coordinate and zero elsewhere, and the other with mean 0.5 in the second coordinate and zero elsewhere.

Our base kernel set $\{k_u\}_{u=1}^d$ contained only $d$ univariate kernels with fixed bandwidth (one for each dimension): in other words, this was a feature selection problem. We used two kernel selection strategies arising from our criterion in (9): *opt* - the kernel from the set $\mathcal{K}$ that maximizes the ratio $\hat{\eta}_k / \hat{\sigma}_{k,\lambda}$, as described in Section 4, and *max-ratio* - the single base kernel $k_u$ with largest $\hat{\eta}_u / \hat{\sigma}_{u,\lambda}$.

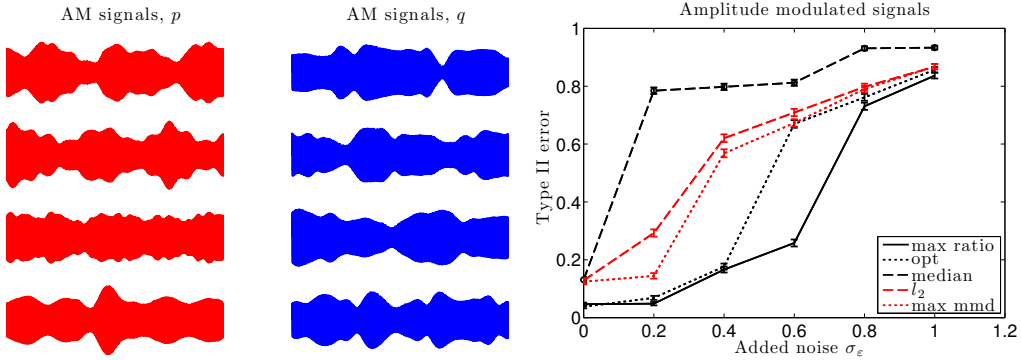

Figure 2: **Left:** amplitude modulated signals, four samples from each of $p$ and $q$ prior to noise being added. **Right**: AM results, Type II error vs added noise, average over 5000 trials, $m = n = 10^4$. The asymptotic test level was $\alpha = 0.05$. Error bars give the 95% Wald confidence interval.

We used $\lambda_n = 10^{-4}$ in both cases. An alternative kernel selection procedure is simply to maximize the MMD on the training data, which is equivalent to minimizing the error in classifying $p$ vs. $q$ under linear loss [15]. In this case, it is necessary to bound the norm of $\beta$, since the test statistic can otherwise be increased without limit by rescaling the $\beta$ entries. We employed two such kernel selection strategies: *max-mmd* - a single base kernel $k_u$ that maximizes $\hat{\eta}_u$ (as proposed in [15]), and $l_2$ - a kernel from the set $\mathcal{K}$ that maximizes $\hat{\eta}_k$ subject to the constraint $\|\beta\|_2 \le 1$ on the vector of weights.

Our results are shown in Figure 1. We see that *opt* and $l_2$ perform much better than *max-ratio* and *max-mmd*, with the former each having large $\hat{\beta}^*$ weights in both the relevant dimensions, whereas the latter are permitted to choose only a single kernel. The performance advantage decreases as more irrelevant dimensions are added. Also note that on these data, there is no statistically significant difference between *opt* and $l_2$, or between *max-ratio* and *max-mmd*.

Difficult problems in two-sample testing arise when the main data variation does not reflect the difference between $p$ and $q$; rather, this is encoded as perturbations at much smaller lengthscales. In these cases, a good choice of kernel becomes crucial. Both remaining experiments are of this type. In the second experiment, $p$ and $q$ were both grids of Gaussians in two dimensions, where $p$ had unit covariance matrices in each mixture component, and $q$ was a grid of correlated Gaussians with a ratio $\varepsilon$ of largest to smallest covariance eigenvalues. A sample dataset is provided in Figure 1. The testing problem becomes more difficult when the number of Gaussian centers in the grid increases, and when $\varepsilon \to 1$. In experiments, we used a five-by-five grid.

We compared *opt, max-ratio*, *max-mmd*, and $l_2$, as well as an additional approach, *xval*, for which we chose the best kernel from $\{k_u\}_{u=1}^d$ by five-fold cross-validation, following [17]. In this case, we learned a witness function on four fifths of the training data, and used it to evaluate the linear loss on $p$ vs $q$ for the rest of the training data (see [7, Section 2.3] for the witness function definition, and [15] for the classification interpretation of the MMD). We made repeated splits to obtain the average validation error, and chose the kernel with the highest average MMD on the validation sets (equivalently, the lowest average linear loss). This procedure has cost $O(m^2)$, and is much more computationally demanding than the remaining approaches.

Our base kernels $\{k_u\}_{u=1}^d$ in (3) were multivariate isotropic Gaussians with bandwidth varying between $2^{-10}$ and $2^{15}$, with a multiplicative step-size of $2^{0.5}$, and we set $\lambda_n = 10^{-5}$. Results are plotted in Figure 1: *opt* and *max-ratio* are statistically indistinguishable, followed in order of decreasing performance by *xval*, *max-mmd*, and $l_2$. The median heuristic fails entirely, yielding the 95% error expected under the null hypothesis. It is notable that the cross-validation approach performs less well than our criterion, which suggests that a direct approach addressing the Type II error is preferable to optimizing the classifier performance.

In our final experiment, the distributions $p, q$ were short samples of amplitude modulated (AM) signals, which were carrier sinusoids with amplitudes scaled by different audio signals for $p$ and $q$.

These signals took the form

$$y(t) = \cos(\omega_c t)\left(As(t) + o_c\right) + n(t),$$

where $y(t)$ is the AM signal at time $t$, $s(t)$ is an audio signal, $\omega_c$ is the frequency of the carrier signal, $A$ is an amplitude scaling parameter, $o_c$ is a constant offset, and $n(t)$ is i.i.d. Gaussian noise with standard deviation $\sigma_\varepsilon$. The source audio signals were [5, Vol. 1, Track 2; Vol. 2 Track 17], and had the same singer but different accompanying instruments. Both songs were normalized to have unit standard deviation, to avoid a trivial distinction on the basis of sound volume. The audio was sampled at 8kHz, the carrier was at 24kHz, and the resulting AM signals were sampled at 120kHz. Further settings were $A = 0.5$ and $o_c = 2$. We extracted signal fragments of length 1000, corresponding to a time duration of $8.3 \times 10^{-3}$ seconds in the original audio. Our base kernels $\{k_u\}_{u=1}^d$ in (3) were multivariate isotropic Gaussians with bandwidth varying between $2^{-15}$ and $2^{15}$, with a multiplicative step-size of 2, and we set $\lambda_n = 10^{-5}$. Sample extracts from each source and Type II error vs noise level $\sigma_\varepsilon$ are shown in Figure 2. Here *max-ratio* does best, with successively decreasing performance by *opt*, *max-mmd*, $l_2$, and *median*. We remark that in the second and third experiments, simply choosing the kernel $k_u$ with largest ratio $\hat{\eta}_u/\hat{\sigma}_{u,\lambda}$ does as well or better than solving for $\hat{\beta}^*$ in (11). The *max-ratio* strategy is thus recommended when a single best kernel exists in the set $\{k_u\}_{u=1}^d$, although it clearly fails when a linear combination of several kernels is needed (as in the first experiment).

Further experiments are provided in the supplementary material. These include an empirical verification that the Type I error is close to the design parameter $\alpha$, and that kernels are not chosen at extreme values when the null hypothesis holds, additional AM experiments, and further synthetic benchmarks.

## 6   Conclusions

We have proposed a criterion to explicitly optimize the Hodges and Lehmann asymptotic relative efficiency for the kernel two-sample test: the kernel parameters are chosen to minimize the asymptotic Type II error at a given Type I error. In experiments using linear combinations of kernels, this approach often performs significantly better than the simple strategy of choosing the kernel with largest MMD (the previous best approach), or maximizing the MMD subject to an $\ell_2$ constraint on the kernel weights, and yields good performance even when the median heuristic fails completely.

A promising next step would be to optimize over the parameters of a single kernel (e.g., over the bandwidth of an RBF kernel). This presents two challenges: first, in proving that a finite sample estimate of the kernel selection criterion converges, which might be possible following [15]; and second, in efficiently optimizing the criterion over the kernel parameter, where we could employ a DC programming [2] or semi-infinite programming [6] approach.

**Acknowledgements:** Part of this work was accomplished when S. B. was visiting the MPI for Intelligent Systems. We thank Samory Kpotufe and Bernhard Schölkopf for helpful discussions.

## Footnotes

[1]This vector is the concatenation of two four-dimensional vectors, and has eight dimensions.

## References

[1] R. Adler and J. Taylor. *Random Fields and Geometry*. Springer, 2007.

[2] Andreas Argyriou, Raphael Hauser, Charles A. Micchelli, and Massimiliano Pontil. A dc-programming algorithm for kernel selection. In *ICML*, pages 41–48, 2006.

[3] P. L. Bartlett and S. Mendelson. Rademacher and Gaussian complexities: Risk bounds and structural results. *Journal of Machine Learning Research*, 3:463–482, 2002.

[4] A. Berlinet and C. Thomas-Agnan. *Reproducing Kernel Hilbert Spaces in Probability and Statistics*. Kluwer, 2004.

[5] Magnetic Fields. 69 love songs. *Merge*, MRG169, 1999.

[6] P. Gehler and S. Nowozin. Infinite kernel learning. Technical Report TR-178, Max Planck Institute for Biological Cybernetics, 2008.

[7] A. Gretton, K. Borgwardt, M. Rasch, B. Schoelkopf, and A. Smola. A kernel two-sample test. *JMLR*, 13:723–773, 2012.

[8] A. Gretton, K. Borgwardt, M. Rasch, B. Schölkopf, and A. J. Smola. A kernel method for the two-sample problem. In *Advances in Neural Information Processing Systems 15*, pages 513–520, Cambridge, MA, 2007. MIT Press.

[9] A. Gretton, K. Fukumizu, Z. Harchaoui, and B. Sriperumbudur. A fast, consistent kernel two-sample test. In *Advances in Neural Information Processing Systems 22*, Red Hook, NY, 2009. Curran Associates Inc.

[10] Z. Harchaoui, F. Bach, and E. Moulines. Testing for homogeneity with kernel Fisher discriminant analysis. In *Advances in Neural Information Processing Systems 20*, pages 609–616. MIT Press, Cambridge, MA, 2008.

[11] R. A. Horn and C. R. Johnson. *Matrix analysis*. Cambridge Univ Press, 1990.

[12] C. McDiarmid. On the method of bounded differences. In *Survey in Combinatorics*, pages 148–188. Cambridge University Press, 1989.

[13] R. Serfling. *Approximation Theorems of Mathematical Statistics*. Wiley, New York, 1980.

[14] A. J. Smola, A. Gretton, L. Song, and B. Schölkopf. A Hilbert space embedding for distributions. In *Proceedings of the International Conference on Algorithmic Learning Theory*, volume 4754, pages 13–31. Springer, 2007.

[15] B. Sriperumbudur, K. Fukumizu, A. Gretton, G. Lanckriet, and B. Schoelkopf. Kernel choice and classifiability for RKHS embeddings of probability distributions. In *Advances in Neural Information Processing Systems 22*, Red Hook, NY, 2009. Curran Associates Inc.

[16] B. Sriperumbudur, A. Gretton, K. Fukumizu, G. Lanckriet, and B. Schölkopf. Hilbert space embeddings and metrics on probability measures. *Journal of Machine Learning Research*, 11:1517–1561, 2010.

[17] M. Sugiyama, T. Suzuki, Y. Itoh, T. Kanamori, and M. Kimura. Least-squares two-sample test. *Neural Networks*, 24(7):735–751, 2011.

[18] A. W. van der Vaart and J. A. Wellner. *Weak Convergence and Empirical Processes*. Springer, 1996.

